# Structured Learning for Cell Tracking

**Xinghua Lou, Fred A. Hamprecht**
Heidelberg Collaboratory for Image Processing (HCI)
Interdisciplinary Center for Scientific Computing (IWR)
University of Heidelberg, Heidelberg 69115, Germany
{xinghua.lou,fred.hamprecht}@iwr.uni-heidelberg.de

## Abstract

We study the problem of learning to track a large quantity of homogeneous objects such as cell tracking in cell culture study and developmental biology. Reliable cell tracking in time-lapse microscopic image sequences is important for modern biomedical research. Existing cell tracking methods are usually kept simple and use only a small number of features to allow for manual parameter tweaking or grid search. We propose a structured learning approach that allows to learn optimum parameters automatically from a training set. This allows for the use of a richer set of features which in turn affords improved tracking compared to recently reported methods on two public benchmark sequences.

## 1   Introduction

One distinguishing property of life is its temporal dynamics, and it is hence only natural that time lapse experiments play a crucial role in current research on signaling pathways, drug discovery and developmental biology [17]. Such experiments yield a very large number of images, and reliable automated cell tracking emerges naturally as a prerequisite for further quantitative analysis.

Even today, cell tracking remains a challenging problem in dense populations, in the presence of complex behavior or when image quality is poor. Existing cell tracking methods can broadly be categorized as deformable models, stochastic filtering and object association. Deformable models combine detection, segmentation and tracking by initializing a set of models (e.g. active contours) in the first frame and updating them in subsequent frames (e.g. [17, 8]). Large displacements are difficult to capture with this class of techniques and are better handled by state space models, e.g. in the guise of stochastic filtering. The latter also allows for sophisticated observation models (e.g. [20]). Stochastic filtering builds on a solid statistical foundation, but is often limited in practice due to its high computational demands. Object association methods approximate and simplify the problem by separating the detection and association steps: once object candidates have been detected and characterized, a second step suggests associations between object candidates at different frames. This class of methods scales well [21, 16, 13] and allows the tracking of thousands of cells in 3D [19].

All of the above approaches contain energy terms whose parameters may be tedious or difficult to adjust. Recently, great efforts have been made to produce better energy terms with helps of machine learning techniques. This was first accomplished by casting tracking as a local affinity prediction problem such as binary classification with either offline [1] or online learning [11, 5, 15], weakly supervised learning with imperfect oracles [27], manifold appearance model learning [25], or ranking [10, 18]. However, these local methods fail to capture the very important dependency among associations, hence the resulting local affinities do not necessarily guarantee a better global association [26]. To address this limitation, [26] extended the RankBoost method from [18] to rank global associations represented as a Conditional Random Field (CRF). Regardless of this, it has two major drawbacks. Firstly, it depends on a set of artificially generated false association samples that can make the training data particularly imbalanced and the training procedure too expensive

for large-scale tracking problems. Secondly, RankBoost desires the ranking feature to be positively correlated with the final ranking (i.e. the association score) [10]. This in turn requires careful pre-adjustment of the sign of each feature based on some prior knowledge [18]. Actually, this prior knowledge may not always be available or reliable in practice.

The contribution of this paper is two-fold. We first present an extended formulation of the object association models proposed in the literature. This generalization improves the expressiveness of the model, but also increases the number of parameters. We hence, secondly, propose to use structured learning to automatically learn optimum parameters from a training set, and hence profit fully from this richer description. Our method addresses the limitations of aforementioned learning approaches in a principled way.

The rest of the paper is organized as follows. In section 2, we present the extended object association models and a structured learning approach for global affinity learning. In section 3, an evaluation shows that our framework inherits the runtime advantage of object association while addressing many of its limitations. Finally, section 4 states our conclusions and discusses future work.

## 2    Structured Learning for Cell Tracking

### 2.1    Association Hypotheses and Scoring

We assume that a previous detection and segmentation step has identified object candidates in all frames, see Fig. 1. We set out to find that set of object associations that best explains these observations. To this end, we admit the following set $E$ of standard events [21, 13]: a cell can *move* or *divide* and it can *appear* or *disappear*. In addition, we allow two cells to (seemingly) *merge*, to account for occlusion or undersegmentation; and a cell can (seemingly) *split*, to allow for the lifting of occlusion or oversegmentation. These additional hypotheses are useful to account for the errors that typically occur in the detection and segmentation step in crowded or noisy data. The distinction between division and split is reasonable given that typical fluorescence stains endow the anaphase with a distinctive appearance.

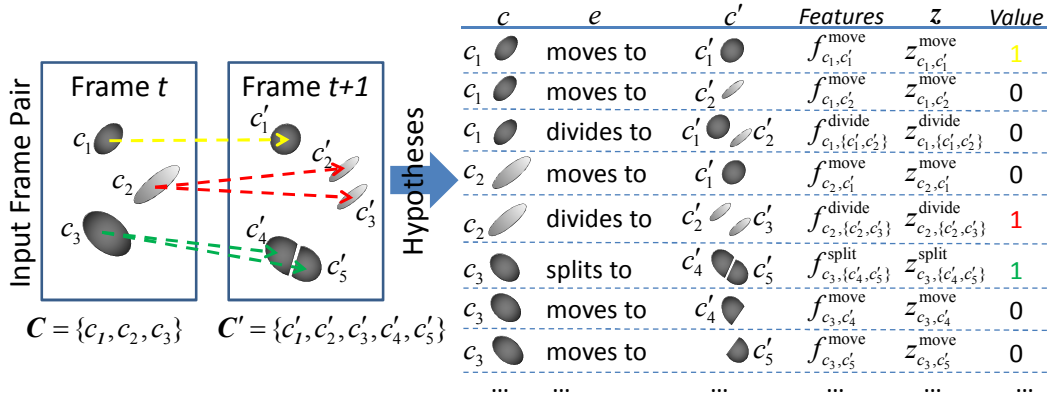

Figure 1: Toy example: two sets of object candidates, and a small subset of the possible association hypotheses. One particular interpretation of the scene is indicated by colored arrows (left) or equivalently by a configuration of binary indicator variables $z$ (rightmost column in table).

Given a pair of object candidate lists $x = \{C, C'\}$ in two neighboring frames, there is a multitude of possible association hypotheses, see Fig. 1. We have two tasks: firstly, to allow only consistent associations (e.g. making sure that each cell in the second frame is accounted for only once); and secondly to identify, among the multitude of consistent hypotheses, the one that is most compatible with the observations, and with what we have learned from the training data.

We express this compatibility of the association between $c \in \mathcal{P}(C)$ and $c' \in \mathcal{P}(C')$ by event $e \in E$ as an inner product $\langle f_{c,c'}^e, w^e \rangle$. Here, $f_{c,c'}^e$ is a feature vector that characterizes the discrepancy (if any) between object candidates $c$ and $c'$; and $w^e$ is a parameter vector that encodes everything we

have learned from the training data. Summing over all object candidates in either of the frames and over all types of events gives the following compatibility function:

$$\mathcal{L}(\boldsymbol{x}, \boldsymbol{z}; \boldsymbol{w}) = \sum_{e \in \boldsymbol{E}} \sum_{c \in \mathcal{P}(\boldsymbol{C})} \sum_{c' \in \mathcal{P}(\boldsymbol{C'})} \langle \boldsymbol{f}_{c,c'}^e, \boldsymbol{w}^e \rangle z_{c,c'}^e \tag{1}$$

$$\text{s. t.} \sum_{e \in \boldsymbol{E}} \sum_{c \in \mathcal{P}(\boldsymbol{C})} z_{c,c'}^e = 1 \text{ and } \sum_{e \in \boldsymbol{E}} \sum_{c' \in \mathcal{P}(\boldsymbol{C'})} z_{c,c'}^e = 1 \text{ with } z_{c,c'}^e \in \{0, 1\} \tag{2}$$

The constraints in the last line involve binary indicator variables $\boldsymbol{z}$ that reflect the consistency requirements: each candidate in the first frame must have a single fate, and each candidate from the second frame a unique history. As an important technical detail, note that $\mathcal{P}(\boldsymbol{C}) := \boldsymbol{C} \cup (\boldsymbol{C} \otimes \boldsymbol{C})$ is a set comprising each object candidate, as well as all ordered pairs of object candidates from a frame[1]. This allows us to conveniently subsume cell divisions, splits and mergers in the above equation. Overall, the compatibility function $\mathcal{L}(\boldsymbol{x}, \boldsymbol{z}; \boldsymbol{w})$, i.e. the global affinity measure, states how well a set of associations $\boldsymbol{z}$ matches the observations $\boldsymbol{f}(\boldsymbol{x})$ computed from the raw data $\boldsymbol{x}$, given the knowledge $\boldsymbol{w}$ from the training set.

The remaining tasks, discussed next, are how to learn the parameters $\boldsymbol{w}$ from the training data (section 2.2); given these, how to find the best possible associations $\boldsymbol{z}$ (section 2.3); and finding useful features (section 2.4).

## 2.2 Structured Max-Margin Parameter Learning

In learning the parameters automatically from a training set, we pursue two goals: first, to go beyond manual parameter tweaking in obtaining the best possible performance; and second, to make the process as facile as possible for the user. This is under the assumption that most experimentalists find it easier to specify what a correct tracking should look like, rather than what value a more-or-less obscure parameter should have.

Given $N$ training frame pairs $\boldsymbol{X} = \{\boldsymbol{x}_n\}$ and their correct associations $\boldsymbol{Z}^* = \{\boldsymbol{z}_n^*\}$, $n = 1, \ldots, N$, the best set of parameters is the optimizer of

$$\arg \min_{\boldsymbol{w}} \mathcal{R}(\boldsymbol{w}; \boldsymbol{X}, \boldsymbol{Z}^*) + \lambda \Omega(\boldsymbol{w}) \tag{3}$$

Here, $\mathcal{R}(\boldsymbol{w}; \boldsymbol{X}, \boldsymbol{Z}^*)$ measures the empirical loss of the current parametrization $\boldsymbol{w}$ given the training data $\boldsymbol{X}, \boldsymbol{Z}^*$. To prevent overfitting to the training data, this is complemented by the regularizer $\Omega(\boldsymbol{w})$ that favors parsimonious models. We use $L_1$ or $L_2$ regularization ($\Omega(\boldsymbol{w}) = ||\boldsymbol{w}||_p^p/p$, $p = \{1, 2\}$), i.e. a measure of the length of the parameter vector $\boldsymbol{w}$. The latter is often used for its numerical efficiency, while the former is popular thanks to its potential to induce sparse solutions (i.e., some parameters can become zero). The empirical loss is given by $\mathcal{R}(\boldsymbol{w}; \boldsymbol{X}, \boldsymbol{Z}^*) = \frac{1}{N} \sum_{i=1}^N \Delta(\boldsymbol{z}_n^*, \hat{\boldsymbol{z}}_n(\boldsymbol{w}; \boldsymbol{x}_n))$. Here $\Delta(\boldsymbol{z}^*, \hat{\boldsymbol{z}})$ is a loss function that measures the discrepancy between a true association $\boldsymbol{z}^*$ and a prediction by specifying the fraction of missed events w.r.t. the ground truth:

$$\Delta(\boldsymbol{z}^*, \hat{\boldsymbol{z}}) = \frac{1}{|\boldsymbol{z}^*|} \sum_{e \in \boldsymbol{E}} \sum_{c \in \mathcal{P}(\boldsymbol{C})} \sum_{c' \in \mathcal{P}(\boldsymbol{C'})} z_{c,c'}^{*e} (1 - \hat{z}_{c,c'}^e). \tag{4}$$

This decomposable function allows for exact inference when solving Eq. 5 [6].

Importantly, both the input (objects from a frame pair) and output (associations between objects) in this learning problem are *structured*. We hence resort to max-margin structured learning [2] to exploit the structure and dependency within the association hypotheses. In comparison to other aforementioned learning methods, structured learning allows us to directly learn the global affinity measure, avoid generating many artificial false association samples, and drop any assumptions on the signs of the features. Structured learning has been successfully applied to many complex real world problems such as word/sequence alignment [22, 24], graph matching [6], static analysis of binary executables [14] and segmentation [3].

In particular, we attempt to find the decision boundary that maximizes the margin between the correct association $\boldsymbol{z}_n^*$ and the closest runner-up solution. An equivalent formulation is the condition

that the score of $\boldsymbol{z}_n^*$ be greater than that of any other solution. To allow for regularization, one can relax this constraint by introducing slack variables $\xi_n$, which finally yields the following objective function for the max-margin structured learning problem from Eq. 3:

$$
\begin{aligned}
\underset{\boldsymbol{w}, \boldsymbol{\xi} \geq \boldsymbol{0}}{\arg\min} \quad & \frac{1}{N} \sum_{n=1}^{N} \xi_n + \lambda \Omega(\boldsymbol{w}) \\
\text{s. t.} \quad & \forall n, \forall \hat{\boldsymbol{z}}_n \in \mathcal{Z}_n : \mathcal{L}(\boldsymbol{x}_n, \boldsymbol{z}_n^*; \boldsymbol{w}) - \mathcal{L}(\boldsymbol{x}_n, \hat{\boldsymbol{z}}_n; \boldsymbol{w}) \geq \Delta(\boldsymbol{z}_n^*, \hat{\boldsymbol{z}}_n) - \xi_n,
\end{aligned}
\tag{5}
$$

where $\mathcal{Z}_n$ is the set of possible consistent associations and $\Delta(\boldsymbol{z}_n^*, \hat{\boldsymbol{z}}_n) - \xi_n$ is known as "margin-rescaling" [24]. Intuitively, it pushes the decision boundary further away from the "bad" solutions with high losses.

## 2.3 Inference and Implementation

Since Eq. 5 involves an exponential number of constraints, the learning problem cannot be represented explicitly, let alone solved directly. We thus resort to the *bundle method* [23] which, in turn, is based on the *cutting-planes* approach [24]. The basic idea is as follows: Start with some parametrization $\boldsymbol{w}$ and no constraints. Iteratively find, first, the optimum associations for the current $\boldsymbol{w}$ by solving, for all $n$, $\hat{\boldsymbol{z}}_n = \arg\max_{\boldsymbol{z}} \{\mathcal{L}(\boldsymbol{x}_n, \boldsymbol{z}; \boldsymbol{w}) + \Delta(\boldsymbol{z}_n^*, \boldsymbol{z})\}$. Use all these $\hat{\boldsymbol{z}}_n$ to identify the most violated constraint, and add it to Eq. 5. Update $\boldsymbol{w}$ by solving Eq. 5 (with added constraints), then find new best associations, etc. pp. For a given parametrization, the optimum associations can be found by integer linear programming (ILP) [16, 21, 13].

Our framework has been implemented in Matlab and C++, including a labeling GUI for the generation of training set associations, feature extraction, model inference and the bundle method. To reduce the search space and eliminate hypotheses with no prospect of being realized, we constrain the hypotheses to a $k$-nearest neighborhood with distance thresholding. We use IBM CPLEX[2] as the underlying optimization platform for the ILP, quadratic programming and linear programming as needed for solving Eq. 5 [23].

## 2.4 Features

To differentiate similar events (e.g. division and split) and resolve ambiguity in model inference, we need rich features to characterize different events. In additional to basic features such as size/position [21] and intensity histogram [16], we also designed new features such as "shape compactness" for oversegmentation and "angle pattern" for division. Shape compactness relates the summed areas of two object candidates to the area of their union's convex hull. Angle pattern describes the constellation of two daughter cells relative to their mother. Features can be defined on a pair of object candidates or on an individual object candidate only. Our features are categorized in Table 1. Note that the same feature can be used for different events.

Table 1: Categorization of features.

|  | Feature Description |
|---|---|
| Position | difference in position, distance to border, overlap with border; |
| Intensity | difference in intensity histogram/sum/mean/deviation, intensity of father cell; |
| Shape | difference in shape, difference in size, shape compactness, shape evenness; |
| Others | division angle pattern, mass evenness, eccentricity of father cell. |

## 3 Results

We evaluated the proposed method on two publicly available image sequences provided in conjunction with the DCellIQ project[3] [16] and the Mitocheck project[4] [12]. The two datasets show a certain degree of variations such as illumination, cell density and image compression artifacts (Fig. 2). The

GFP stained cell nuclei were segmented using the method in [19], yielding an F-measure over 99.3% by counting. Full ground truth associations for training and evaluation were generated with a Matlab GUI tool at a rate of approximately 20 frames/hour. Some statistics about these two datasets are shown in Table 2.

Table 2: Some statistics about the datasets in our evaluation.

| Name | Image Size | No. of Frames | No. of Cells | Segm. F-Measure | Compressed |
|---|---|---|---|---|---|
| DCellIQ | $512 \times 672$ | 100 | 10664 | 99.5% | No |
| Mitocheck | $1024 \times 1344$ | 94 | 24096 | 99.3% | Yes |

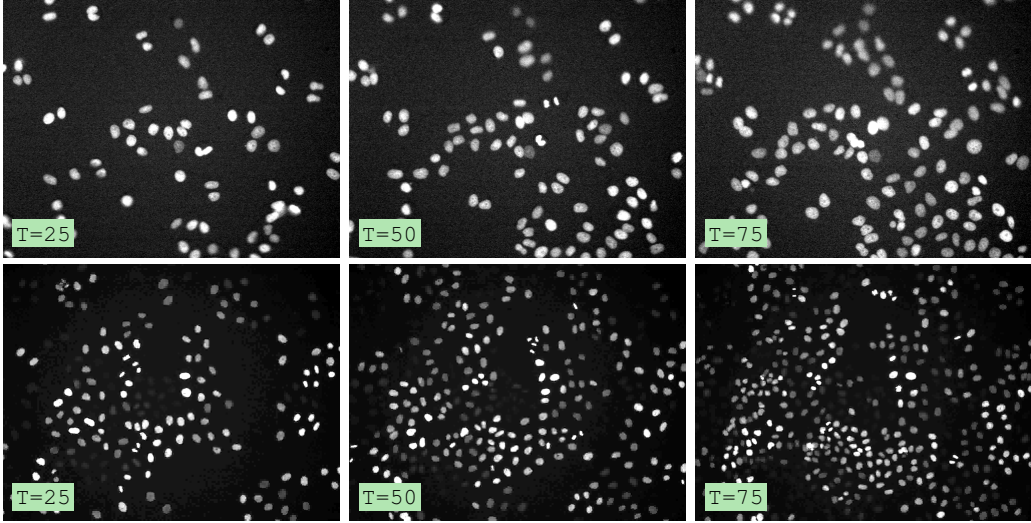

Figure 2: Selected raw images from the DCellIQ sequence (top) and the Mitocheck sequence (bottom). The Mitocheck sequence exhibits higher cell density, larger intensity variability and "blockness" artifacts due to image compression.

**Task 1: Efficient Tracking for a Given Sequence**

We first evaluate our method on a task that is frequently encountered in practice: the user simply wishes to obtain a good tracking for a given sequence with the smallest possible effort. For a fair comparison, we extended Padfield's method [21] to account for the six events described in section 2.1 and used the same features (viz., size and position) and weights as in [21]. Hand-tuning of the parameters results in a high accuracy of 98.4% (i.e. 1 - total loss) as shown in Table 3 (2nd row). A detailed analysis of the error counts for specific events shows that the method accounts well for moves, but has difficulty with disappearance and split events. This is mainly due to the limited descriptive power of the simple features used. To study the difference between manual tweaking and learning of the parameters, we used the learning framework presented here to optimize the model and obtained a reduction of the total loss from 1.64% to 0.65% (3rd row). This can be considered as the limit of this model. Note that the learned parametrization actually deteriorates the detection of divisions because the learning aims at minimizing the overall loss across all events. In obtaining these results, one third of the entire sequence was used for training, just as in all subsequent comparisons.

With 37 features included and their weights optimized using structured learning, our model fully profits from this richer description and achieves a total loss of only 0.30% (4th row) which is a significant improvement over [21, 16] (2nd/7th row) and manual tweaking (6th row). Though a certain amount of efforts is needed for creating the training set, our method allows experimentalists to contribute their expertise in an intuitive fashion. Some example associations are shown in Fig. 3.

The learned parameters are summarized in Fig. 4 (top). They afford the following observations: Firstly, features on cell size and shape are generally of high importance, which is in line with the assumption in [21]. Secondly, the correlations of the features with the final association score are

Table 3: Performance comparison on the DCellIQ dataset. The header row shows the number of events occurring for moves, divisions, appearance, disappearance, splits and mergers. The remaining entries give the error counts for each event, summed over the entire sequence.

| | mov | div | app | dis | spl | mer | total loss |
| | 10156 | 104 | 78 | 76 | 54 | 55 | - |
|---|---|---|---|---|---|---|---|
| Padfield *et al.* [21] | 71 | 18 | 16 | 26 | 30 | 12 | 1.64% |
| Padfield *et al.* w/ learning | 21 | 25 | 5 | 5 | 6 | 10 | 0.65% |
| Ours w/ learning ($L_2$ regula.) | **15** | **6** | **4** | 1 | **2** | 6 | **0.30%** |
| Ours w/ learning ($L_1$ regula.) | 22 | **6** | 9 | 3 | 4 | 9 | 0.45% |
| Ours w/ manual tweaking | 56 | 24 | 16 | 19 | 2 | **5** | 1.12% |
| Li *et al.* [16] | - | - | - | - | - | - | 6.18%[a] |
| Local learning by Random Forest | 18 | 14 | 2 | **0** | 12 | 13 | 0.55% |

[a]Here we use the best reported error matching rate in [16] (similar to our loss).

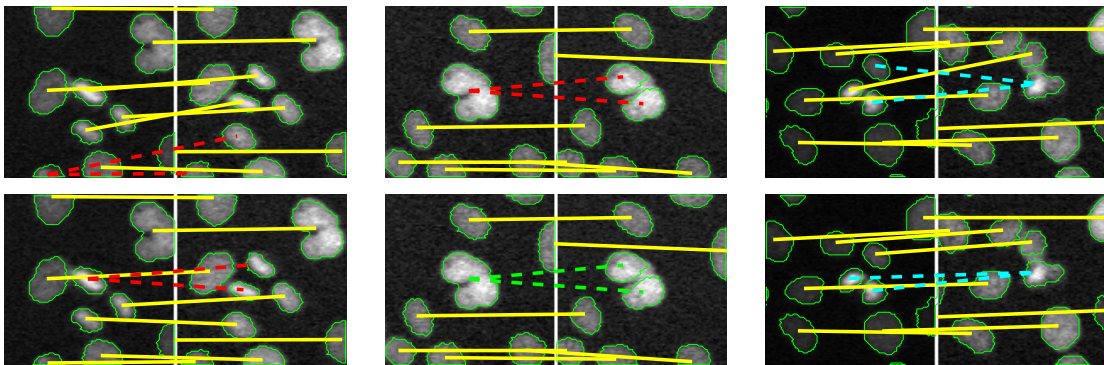

Figure 3: Some diverging associations by [21] (top) and our method (bottom). Color code: yellow – move; red – division; green – split; cyan – merger.

automatically learned. For example, shape compactness is positively correlated with split but negatively with division. This is in line with the intuition that an oversegmentation conserves compact shape, while a true division seemingly pushes the daughters far away from each other (in the present kind of data, where only DNA is labeled). Finally, in spite of the regularization, many features are associated with large parameter values, which is key to the improved expressive power.

**Task 2: Tracking for High-Throughput Experiments**

The experiment described in the foregoing draws both training and test samples from the same time lapse experiment. However, in high-throughput experiments such as in the Mitocheck project [12], it is more desirable to train on one or a few sequences, and make predictions on many others. To emulate this situation, we have used the parameters $w$ trained in the foregoing on the DCellIQ sequence [16] and used these to estimate the tracking of the Mitocheck dataset. The main focus of the Mitocheck project is on accurate detection of mitosis (cell division). Despite the difference in illumination and cell density from the training data, and despite the segmentation artifacts caused by the compression of the image sequence, our method shows a high generalization capability and obtains a total loss of 0.78%. In particular, we extract 93.2% of 384 mitosis events which is a significant improvement over the mitosis detection rate reported in [12] (81.5%, 294 events).

**Comparison to Local Affinity Learning**

We also developed a local affinity learning approach that is in spirit of [1, 15]. Rather than using AdaBoost [9], we chose Random Forest (RF) [4] which provides fairly comparable classification power [7]. We sample positive associations from the ground truth and randomly generate false associations. RF classifiers are built for each event *independently*. The predicted probabilities by the RF classifiers are used to compute the overall association score as in Eq. 6 (with the same constraints in Eq. 2). Since we have multiple competing events (one cell can only have a single

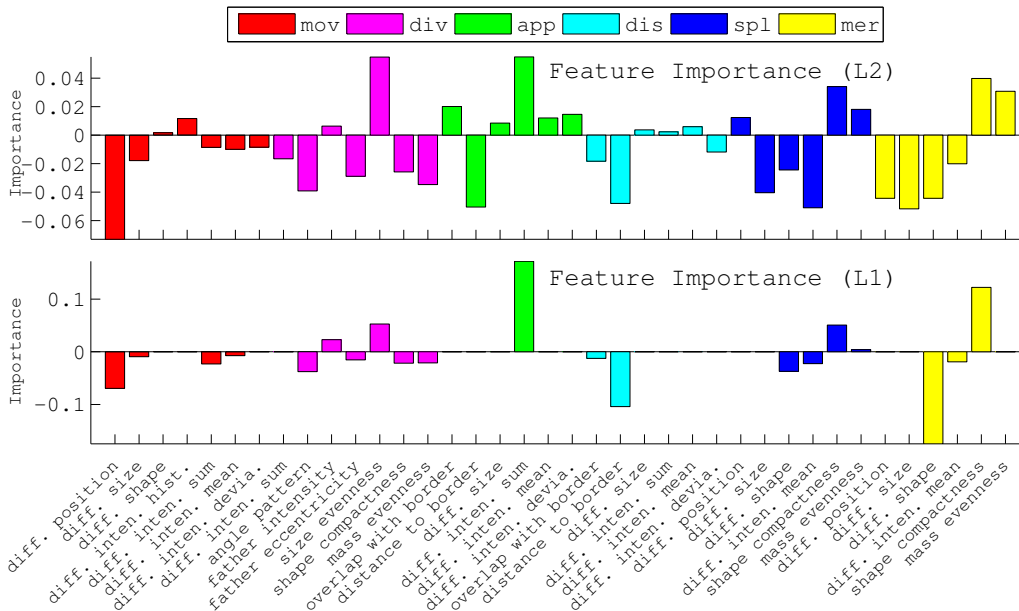

Figure 4: Parameters $\boldsymbol{w}$ learned from the training data with $L_2$ (top) or $L_1$ (bottom) regularization. Parameters weighing the features for different events are colored differently. Both parameter vectors are normalized to unit 1-norm, i.e. $\|\boldsymbol{w}\|_1 = 1$.

Table 4: Performance comparison on the Mitocheck dataset. The method was trained on the DCellIQ dataset. The header row shows the number of events occurring for moves, divisions, appearance, disappearance, splits and mergers. The remaining entries give the error counts for each event, summed over the entire sequence.

|  | mov | div | app | dis | spl | mer | total loss |
|---|---|---|---|---|---|---|---|
|  | 22520 | 384 | 310 | 304 | 127 | 132 | - |
| Padfield *et al.* w/ learning | 171 | 85 | 58 | 47 | 53 | 13 | 1.39% |
| Ours w/ learning ($L_2$ regula.) | 98 | **26** | **31** | **25** | 43 | **9** | **0.78%** |
| Ours w/ learning ($L_1$ regula.) | **93** | 35 | 54 | 25 | **26** | 48 | 0.98% |
| Local learning by Random Forest | 214 | 281 | 162 | **10** | 82 | 68 | 2.33% |

fate), we also introduce weights $\{\alpha_e\}$ to capture the dependencies between events. These weights are optimized via a grid search on the training data.

$$\mathcal{L}(\boldsymbol{x}, \boldsymbol{z}; \boldsymbol{w}) = \sum_{e \in \boldsymbol{E}} \sum_{c \in \mathcal{P}(\boldsymbol{C})} \sum_{c' \in \mathcal{P}(\boldsymbol{C'})} \alpha^e \mathrm{Prob}(\boldsymbol{f}^e_{c,c'}) z^e_{c,c'} \tag{6}$$

The results are shown in Table 3 (8th row) and Table 4 (5th row), which afford the following observations. Firstly, a locally strong affinity prediction does not necessarily guarantee a better global association. Secondly, local learning shows particularly weak generalization capability.

**Sensitivity to Training Set**

The success of supervised learning depends on the representativeness (and hence also size) of the training set. To test the sensitivity of the results to the training data used, we drew different numbers of training image pairs randomly from the entire sequence and used the remaining pairs for testing. For each training set size, this experiment is repeated 10 times. The mean and deviation of the losses on the respective test sets is shown in Fig. 5. According to the one-standard-error-rule, associations between at least 15 or 20 image pairs are desirable, which can be accomplished in well below an hour of annotation work.

**$L_1$ vs. $L_2$ Regularization**

The results of $L_1$ vs. $L_2$ regularization are comparable (see Table 3 and Table 4). While $L_1$ regularization yields sparser feature selection 4 (bottom), it has a much slower convergence rate (Fig. 6). The staircase structure shows that, due to sparse feature selection, the bundle method has to find more constraints to escape from a local minimum.

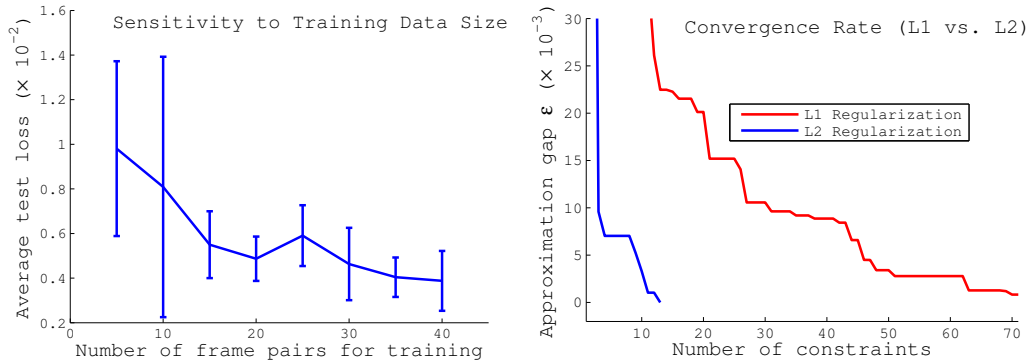

Figure 5: Learning curve of structured learning (with $L_2$ regularization).

Figure 6: Convergence rates of structured learning ($L_1$ vs. $L_2$ regularization).

## 4 Conclusion & Future Work

We present a new cell tracking scheme that uses more expressive features and comes with a structured learning framework to train the larger number of parameters involved. Comparison to related methods shows that this learning scheme brings significant improvements in performance and, in our opinion, usability.

We currently work on further improvement of the tracking by considering more than two frames at a time, and on an active learning scheme that should reduce the amount of required training inputs.

## Acknowledgement

We are very grateful for partial financial support by CellNetworks Cluster (EXC81), FORSYS-ViroQuant (0313923), SBCancer, DFG (GRK 1653) and "Enable fund" of University of Heidelberg. We also thank Bjoern Andres, Jing Yuan and Christoph Straehle for their comments on the manuscript.

## Footnotes

[1]For the example in Fig. 1, $\mathcal{P}(\boldsymbol{C}) = \{c_1, c_2, c_3, \{c_1, c_2\}, \{c_1, c_3\}, \{c_2, c_3\}\}$.

[2]http://www-01.ibm.com/software/integration/optimization/cplex-optimizer/

[3]http://www.cbi-tmhs.org/Dcelliq/files/051606_HeLaMCF10A_DMSO_1.rar

[4]http://www.mitocheck.org/cgi-bin/mtc?action=show_movie;query=243867

## References

[1] S. Avidan. Ensemble Tracking. In *CVPR*, 2005.

[2] G. Bakir, T. Hofmann, B. Schoelkopf, A. J. Smola, B. Taskar, and S. Vishwanathan. *Predicting Structured Data*. MIT Press, Cambridge, MA, 2006.

[3] L. Bertelli, T. Yu, D. Vu, and B. Gokturk. Kernelized Structural SVM Learning for Supervised Object Segmentation. In *CVPR*, 2011.

[4] L. Breiman. Random Forests. *Mach Learn*, 45(1):5–32, 2001.

[5] M. D. Breitenstein, F. Reichlin, B. Leibe, E. Koller-Meier, and L. V. Gool. Robust Tracking-by-Detection using a Detector Confidence Particle Filter. In *ICCV*, 2009.

[6] T. S. Caetano, J. J. McAuley, L. Cheng, Q. V. Le, and A. J. Smola. Learning Graph Matching. *IEEE T Pattern Anal*, 31(6):1048–1058, 2009.

[7] R. Caruana and A. Niculescu-Mizil. An Empirical Comparison of Supervised Learning Algorithms. In *ICML*, pages 161–168, 2006.

[8] O. Dzyubachyk, W. A. van Cappellen, J. Essers, et al. Advanced Level-Set-Based Cell Tracking in Time-Lapse Fluorescence Microscopy. *IEEE T Med Imag*, 29(3):852, 2010.

[9] Y. Freund. An adaptive version of the boost by majority algorithm. *Mach Learn*, 43(3):293–318, 2001.

[10] Y. Freund, R. Iyer, R. E. Schapire, , and Y. Singer. An Efficient Boosting Algorithm for Combining Preferences. *J Mach Learn Res*, 4:933–969, 2003.

[11] H. Grabner and H. Bischof. On-line Boosting and Vision. In *CVPR*, 2006.

[12] M. Held, M. H. A. Schmitz, et al. CellCognition: time-resolved phenotype annotation in high-throughput live cell imaging. *Nature Methods*, 7(9):747–754, 2010.

[13] T. Kanade, Z. Yin, R. Bise, S. Huh, S. E. Eom, M. Sandbothe, and M. Chen. Cell Image Analysis: Algorithms, System and Applications. In *WACV*, 2011.

[14] N. Karampatziakis. Static Analysis of Binary Executables Using Structural SVMs. In *NIPS*, 2010.

[15] C.-H. Kuo, C. Huang, , and R. Nevatia. Multi-Target Tracking by On-Line Learned Discriminative Appearance Models. In *CVPR*, 2010.

[16] F. Li, X. Zhou, J. Ma, and S. Wong. Multiple Nuclei Tracking Using Integer Programming for Quantitative Cancer Cell Cycle Analysis. *IEEE T Med Imag*, 29(1):96, 2010.

[17] K. Li, E. D. Miller, M. Chen, et al. Cell population tracking and lineage construction with spatiotemporal context. *Med Image Anal*, 12(5):546–566, 2008.

[18] Y. Li, C. Huang, and R. Nevatia. Learning to Associate: HybridBoosted Multi-Target Tracker for Crowded Scene. *CVPR*, 2009.

[19] X. Lou, F. O. Kaster, M. S. Lindner, et al. DELTR: Digital Embryo Lineage Tree Reconstructor. In *ISBI*, 2011.

[20] E. Meijering, O. Dzyubachyk, I. Smal, and W. A. van Cappellen. Tracking in cell and developmental biology. *Semin Cell Dev Biol*, 20(8):894 – 902, 2009.

[21] D. Padfield, J. Rittscher, and B. Roysam. Coupled Minimum-Cost Flow Cell Tracking for High-Throughput Quantitative Analysis. *Med Image Anal*, 2010.

[22] B. Taskar, S. Lacoste-Julien, and M. I. Jordan. Structured Prediction, Dual Extragradient and Bregman Projections. *J Mach Learn Res*, 7:1627–1653, 2006.

[23] C. H. Teo, S. V. N. Vishwanthan, A. J. Smola, and Q. V. Le. Bundle methods for regularized risk minimization. *J Mach Learn Res*, 11:311–365, 2010.

[24] I. Tsochantaridis, T. Joachims, T. Hofmann, and Y. Altun. Large Margin Methods for Structured and Interdependent Output Variables. *J Mach Learn Res*, 6(2):1453, 2006.

[25] X. Wang, G. Hua, and T. X. Han. Discriminative Tracking by Metric Learning. In *ECCV*, 2010.

[26] B. Yang, C. Huang, and R. Nevatia. Learning Affinities and Dependencies for Multi-Target Tracking using a CRF Model. In *CVPR*, 2011.

[27] B. Zhong, H. Yao, S. Chen, et al. Visual Tracking via Weakly Supervised Learning from Multiple Imperfect Oracles. In *CVPR*, 2010.

